# Sequential effects: Superstition or rational behavior?

**Angela J. Yu**
Department of Cognitive Science
University of California, San Diego
ajyu@ucsd.edu

**Jonathan D. Cohen**
Department of Psychology
Princeton University
jdc@princeton.edu

## Abstract

In a variety of behavioral tasks, subjects exhibit an automatic and apparently sub-optimal *sequential effect*: they respond more rapidly and accurately to a stimulus if it reinforces a local pattern in stimulus history, such as a string of repetitions or alternations, compared to when it violates such a pattern. This is often the case even if the local trends arise by chance in the context of a randomized design, such that stimulus history has no real predictive power. In this work, we use a normative Bayesian framework to examine the hypothesis that such idiosyncrasies may reflect the inadvertent engagement of mechanisms critical for adapting to a changing environment. We show that prior belief in non-stationarity can induce experimentally observed sequential effects in an otherwise Bayes-optimal algorithm. The Bayesian algorithm is shown to be well approximated by linear-exponential filtering of past observations, a feature also apparent in the behavioral data. We derive an explicit relationship between the parameters and computations of the exact Bayesian algorithm and those of the approximate linear-exponential filter. Since the latter is equivalent to a leaky-integration process, a commonly used model of neuronal dynamics underlying perceptual decision-making and trial-to-trial dependencies, our model provides a principled account of *why* such dynamics are useful. We also show that parameter-tuning of the leaky-integration process is possible, using stochastic gradient descent based only on the noisy binary inputs. This is a proof of concept that not only can neurons implement near-optimal prediction based on standard neuronal dynamics, but that they can also learn to tune the processing parameters without explicitly representing probabilities.

## 1 Introduction

One common error human subjects make in statistical inference is that they detect hidden patterns and causes in what are genuinely random data. Superstitious behavior, or the inappropriate linking of stimuli or actions with consequences, can often arise in such situations, something also observed in non-human subjects [1, 2]. One common example in psychology experiments is that despite a randomized experimental design, which deliberately de-correlate stimuli from trial to trial, subjects pick up transient patterns such as runs of *repetitions* and *alternations*, and their responses are facilitated when a stimulus continues to follow a local pattern, and impeded when such a pattern is violated [3]. It has been observed in numerous experiments [3–5], that subjects respond more accurately and rapidly if a trial is consistent with the recent pattern (e.g. $AAAA$ followed by $A$, $BABA$ followed by $B$), than if it is inconsistent (e.g. $AAAA$ followed by $B$, $BABA$ followed by $A$). This *sequential effect* is more prominent when the preceding run has lasted longer. Figure 1a shows reaction time (RT) data from one such experiment [5]. Error rates follow a similar pattern, reflecting a true expectancy-based effect, rather than a shift in RT-accuracy trade-off.

A natural interpretation of these results is that local patterns lead subjects to expect a stimulus, whether explicitly or implicitly. They readily respond when a subsequent stimulus extends the local pattern, and are "surprised" and respond less rapidly and accurately when a subsequent stimulus violates the pattern. When such local patterns persist longer, the subjects have greater confidence in

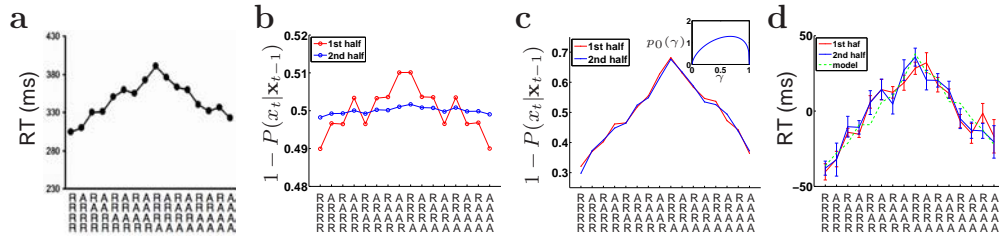

Figure 1: Bayesian modeling of sequential effects. (a) Median reaction time (RT) from Cho et al (2002) affected by recent history of stimuli, in which subjects are required to discriminate a small "o" from a large "O" using button-presses. Along the abscissa are all possible four-trial sub-sequences, in terms of repetitions ($R$) and alternations ($A$). Each sequence, read from top to bottom, proceeds from the earliest stimulus progressively toward the present stimulus. As the effects were symmetric across the two stimulus types, $A$ and $B$, each bin contains data from a pair of conditions (e.g. $RRAR$ can be $AAABB$ or $BBBAA$). RT was fastest when a pattern is reinforced ($RRR$ followed by $R$, or $AAA$ followed by $A$); it is slowest when an "established" pattern is violated ($RRR$ followed by $A$, or $AAA$ followed by $R$). (b) Assuming RT decreases with predicted stimulus probability (i.e. RT increases with $1 - P(x_t|\mathbf{x}_{t-1})$, where $x_t$ is the actual stimulus seen), then FBM would predict much weaker sequential effects in the second half (blue: 720 simulated trials) than in the first half (red: 840 trials). (c) DBM predicts persistently strong sequential effects in both the first half (red: 840 trials) and second half (blue: 720 trials). Inset shows prior over $\gamma$ used; the same prior was also used for the FBM in (b). $\alpha = .77$. (d) Sequential effects in behavioral data were equally strong in the first half (red: 7 blocks of 120 trials each) and the second half (blue: 6 blocks of 120 trials each). Green dashed line shows a linear transformation from the DBM prediction in probability space of (c) into the RT space. The fit is very good given the errorbars (SEM) in the data.

the pattern, and are therefore more surprised and more strongly affected when the pattern is violated. While such a strategy seems plausible, it is also sub-optimal. The experimental design consists of randomized stimuli, thus all runs of repetitions or alternations are spurious, and any behavioral tendencies driven by such patterns are useless. However, compared to artificial experimental settings, truly random sequential events may be rare in the natural environment, where the laws of physics and biology dictate that both external entities and the observer's viewpoint undergo continuous transformations for the most part, leading to statistical regularities that persist over time on characteristic timescales. The brain may be primed to extract such statistical regularities, leading to what appears to be superstitious behavior in an artificially randomized experimental setting.

In section 2, we use Bayesian probability theory to build formally rigorous models for predicting stimuli based on previous observations, and compare differentially complex models to subjects' actual behavior. Our analyses imply that subjects assume statistical contingencies in the task to persist over several trials but *non-stationary* on a longer time-scale, as opposed to being unknown but *fixed* throughout the experiment. We are also interested in understanding how the computations necessary for prediction and learning can be implemented by the neural hardware. In section 3, we show that the Bayes-optimal learning and prediction algorithm is well approximated by a linear filter that weighs past observations exponentially, a computationally simpler algorithm that also seems to fit human behavior. Such an exponential linear filter can be implemented by standard models of neuronal dynamics. We derive an explicit relationship between the assumed rate of change in the world and the time constant of the optimal exponential linear filter. Finally, in section 4, we will show that meta-learning about the rate of change in the world can be implemented by stochastic gradient descent, and compare this algorithm with exact Bayesian learning.

## 2    Bayesian prediction in fixed and changing worlds

One simple internal model that subjects may have about the nature of the stimulus sequence in a 2-alternative forced choice (2AFC) task is that the statistical contingencies in the task remain fixed throughout the experiment. Specifically, they may believe that the experiment is designed such that there is a fixed probability $\gamma$, throughout the experiment, of encountering a repetition ($x_t = 1$) on any given trial $t$ (thus probability $1-\gamma$ of seeing an alternation $x_t = 0$). What they would then learn

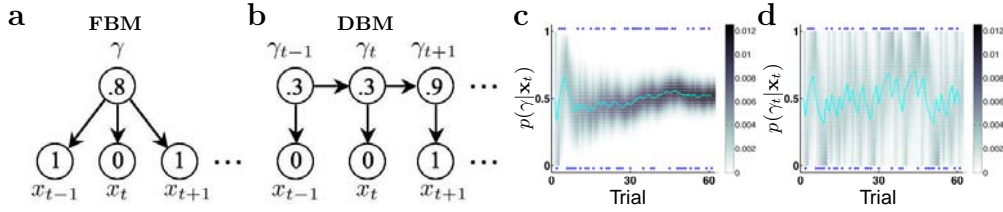

Figure 2: Bayesian inference assuming fixed and changing Bernoulli parameters. (a) Graphical model for the FBM. $\gamma \in [0, 1]$, $x_t \in \{0, 1\}$. The numbers in circles show example values for the variables. (b) Graphical model for the DBM. $\gamma_t = \alpha\delta(\gamma_t - \gamma_{t-1}) + (1 - \alpha)p_0(\gamma_t)$, where we assume the prior $p_0$ to be a Beta distribution. The numbers in circles show examples values for the variables. (c) Grayscale shows the evolution of posterior probability mass over $\gamma$ for FBM (darker color indicate concentration of mass), given the sequence of truly random ($P(x_t) = .5$) binary data (blue dots). The mean of the distribution, in cyan, is also the predicted stimulus probability: $P(x_t = 1|\mathbf{x}_{t-1}) = \langle\gamma|\mathbf{x}_{t-1}\rangle$. (d) Evolution of posterior probability mass for the DBM (grayscale) and predictive probability $P(x_t = 1|\mathbf{x}_{t-1})$ (cyan); they perpetually fluctuate with transient runs of repetitions or alternations.

about the task over the time course of the experiment is the appropriate value of $\gamma$. We call this the Fixed Belief Model (FBM). Bayes' Rule tells us how to compute the posterior:

$$p(\gamma|\mathbf{x}_t) \propto P(\mathbf{x}_t|\gamma)p(\gamma) = \gamma^{r_t+a+1}(1 - \gamma)^{t-r_t+b+1}$$

where $r_t$ denotes the number of repetitions observed so far (up to $t$), $\mathbf{x}_t$ is the set of binary observations $(x_1, \ldots, x_t)$, and the prior distribution $p(\gamma)$ is assumed to be a beta distribution: $p(\gamma) = p_0(\gamma) = Beta(a, b)$. The predicted probability of seeing a repetition on the next trial is the mean of this posterior distribution: $P(x_{t+1} = 1|\mathbf{x}_t) = \int \gamma p(\gamma|\mathbf{x}_t)d\gamma = \langle\gamma|\mathbf{x}_t\rangle$.

A more complex internal model that subjects may entertain is that the relative frequency of repetition (versus alternation) can undergo discrete changes at unsignaled times during the experimental session, such that repetitions are more prominent at times, and alternation more prominent at other times. We call this the Dynamic Belief Model (DBM), in which $\gamma_t$ has a Markovian dependence on $\gamma_{t-1}$, so that with probability $\alpha$, $\gamma_t = \gamma_{t-1}$, and probability $1 - \alpha$, $\gamma_t$ is redrawn from a fixed distribution $p_0(\gamma_t)$ (same Beta distribution as for the prior). The observation $x_t$ is still assumed to be drawn from a Bernoulli process with rate parameter $\gamma_t$. Stimulus predictive probability is now the mean of the iterative prior, $P(x_t = 1|\mathbf{x}_{t-1}) = \langle\gamma_t|\mathbf{x}_{t-1}\rangle$, where

$$\begin{aligned} p(\gamma_t &= \gamma|\mathbf{x}_{t-1}) = \alpha p(\gamma_{t-1} = \gamma|\mathbf{x}_{t-1}) + (1 - \alpha)p_0(\gamma_t = \gamma) \\ p(\gamma_t|\mathbf{x}_t) &\propto P(x_t|\gamma_t)p(\gamma_t|\mathbf{x}_{t-1}) \end{aligned}$$

Figures 2a;b illustrate the two graphical models. Figures 2c;d demonstrate how the two models respond differently to the exact same sequence of truly random binary observations ($\gamma = .5$). While inference in FBM leads to less variable and more accurate estimate of the underlying bias as the number of samples increases, inference in DBM is perpetually driven by local transients. Relating back to the experimental data, we plot the probability of *not* observing the current stimulus for each type of 5-stimulus sequences in Figure 1 for (b) FBM and (c) DBM, since RT is known to lengthen with reduced stimulus expectancy. Comparing the first half of a simulated experimental session (red) with the second half (blue), matched to the number of trials for each subject, we see that sequential effects significantly diminish in the FBM, but persist in the DBM. A re-analysis of the experimental data (Figure 1d) shows that sequential effects also persist in human behavior, confirming that Bayesian prediction based on a (Markovian) changeable world can account for behavioral data, while that based on a fixed world cannot. In Figure 1d, the green dashed line shows that a linear transformation of the DBM sequential effect (from Figure 1c) is quite a good fit of the behavioral data. It is also worth noting that in the behavioral data there is a slight over all preference (shorter RT) for repetition trials. This is easily captured by the DBM by assuming $p_0(\gamma_t)$ to be skewed toward repetitions (see Figure 1c inset). The same skewed prior cannot produce a bias in the FBM, however, because the prior only figures into Bayesian inference once at the outset, and is very quickly overwhelmed by the accumulating observations.

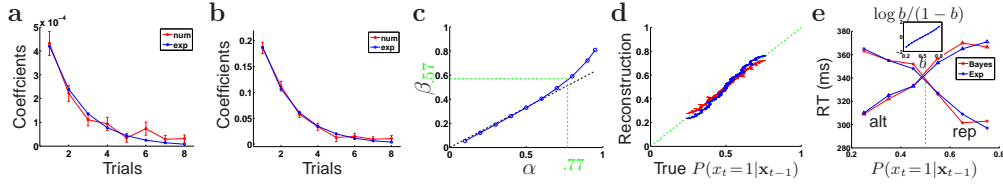

Figure 3: Exponential discounting a good descriptive and normative model. (a) For each of the six subjects, we regressed RR on repetition trials against past observations, $RT \approx C + b_1 x_{t-1} + b_2 x_{t-2} + \dots$, where $x_\tau$ is assigned 0 if it was repetition, and 1 if alternation, the idea being that recent repetition trials should increase expectation of repetition and decrease RR, and recent alternation should decrease expectation of repetition and increase RR on a repetition trial. Separately we also regressed RR's on alternation trials against past observations (assigning 0 to alternation trials, and 1 to repetitions). The two sets of coefficients did not differ significantly and were averaged togther (red: average across subjects, error bars: SEM). Blue line shows the best exponential fit to these coefficients. (b) We regressed $P_t$ obtained from exact Bayesian DBM inference, against past observations, and obtained a set of average coefficients (red); blue is the best exponential fit. (c) For different values of $\alpha$, we repeat the process in (b) and obtain the best exponential decay parameter $\beta$ (blue). Optimal $\beta$ closely tracks the $2/3$ rule for a large range of values of $\alpha$. $\beta$ is .57 in (a), so $\alpha = .77$ was used to generate (b). (d) Both the optimal exponential fit (red) and the $2/3$ rule (blue) approximate the true Bayesian $P_t$ well (green dashed line shows perfect match). $\alpha = .77$. For smaller values of $\alpha$, the fit is even better; for larger $\alpha$, the exponential approximation deteriorates (not shown). (e) For repetition trials, the greater the predicted probability of seeing a repetition ($x_t = 1$), the faster the RT, whether trials are categorized by Bayesian predictive probabilities (red: $\alpha = .77, p_0 = Beta(1.6, 1.3)$), or by linear exponential filtering (blue). For alternation trials, RT's increase with increasing predicted probability of seeing a repetition. Inset: for the biases $b \in [.2, .8]$, the log prior ratio (shift in the initial starting point, and therefore change in the distance to decision boundary) is approximately linear.

## 3 Exponential filtering both normative and descriptive

While Bayes' Rule tells us in theory what the computations ought to be, the neural hardware may only implement a simpler approximation. One potential approximation is suggested by related work showing that monkeys' choices, when tracking reward contingencies that change at unsignaled times, depend linearly on previous observations that are discounted approximately exponentially into the past [6]. This task explicitly examines subjects' ability to track unsignaled statistical regularities, much like the kind we hypothesize to be engaged inadvertently in sequential effects.

First, we regressed the subjects' *reward rate* (RR) against past observations and saw that the linear coefficients decay approximately exponentially into the past (Figure 3a). We define *reward rate* as mean accuracy/mean RT, averaged across subjects; we thus take into account both effects in RT and accuracy as a function of past experiences. We next examined whether there is also an element of exponential discounting embedded in the DBM inference algorithm. Linear regression of the predictive probability $P_t \triangleq P(x_t = 1 | \mathbf{x}_{t-1})$, which should correlate positively with RR (since it correlates positively with accuracy and negatively with RT) against previous observations $x_{t-1}, x_{t-2}, \dots$ yields coefficients that also decay exponentially into the past (Figure 3b): $P_t \approx C + \eta \sum_{\tau=1}^{t-1} \beta^\tau x_{t-\tau}$. Linear exponential filtering thus appears to be both a good descriptive model of behavior, and a good normative model approximating Bayesian inference.

An obvious question is how this linear exponential filter relates to exact Bayesian inference, in particular how the rate of decay relates to the assumed rate of change in the world (parameterized by $\alpha$). We first note that the linear exponential filter has an equivalent iterative form:

$$P_t \triangleq P(x_t = 1 | \mathbf{x}_{t-1}) = C + \eta \sum_{\tau=1}^{t-1} \beta^\tau x_{t-\tau} = C(1 - \beta) + \eta \beta x_{t-1} + \beta P_{t-1} .$$

We then note that the nonlinear Bayesian update rule can also be written as:

$$P_{t+1} = \frac{1}{2}(1 - \alpha) + x_{t-1}\alpha \frac{K_t - P_t^2}{P_t - P_t^2} + \alpha P_t \frac{1 - \frac{K_t}{P_t}}{1 - P_t} \approx \frac{1}{2}(1 - \alpha) + \frac{1}{3}\alpha x_t + \frac{2}{3}\alpha P_t \qquad (1)$$

where $K_t \triangleq \langle \gamma_t^2 | \mathbf{x}_{-1} \rangle$, and we approximate $P_t$ by its mean value $\langle P_t \rangle = 1/2$, and $K_t$ by its mean value $\langle K_t \rangle = 1/3$. These expected values are obtained by expanding $P_t$ and $K_t$ in their iterative forms and assuming $\langle P_t \rangle = \langle P_{t-1} \rangle$ and $\langle K_t \rangle = \langle K_{t-1} \rangle$, and also assuming that $p_0$ is the uniform distribution. We verified numerically (data not shown) that this mean approximation is quite good for a large range of $\alpha$ (though it gets progressively worse when $\alpha \approx 1$, probably because the equilibrium assumptions deviate farther from reality as changes become increasingly rare).

Notably, our calculations imply $\beta \approx \frac{2}{3}\alpha$, which makes intuitive sense, since slower changes should result in longer integration time window, whereas faster changes should result in shorter memory. Figure 3c shows that the best numerically obtained $\beta$ (by fitting an exponential to the linear regression coefficients) for different values of $\alpha$ (blue) is well approximated by the $2/3$ rule (black dashed line). For the behavioral data in Figure 3a, $\beta$ was found to be .57, which implies $\alpha = .77$; the simulated data in Figure 3b are in fact obtained by assuming $\alpha = .77$, hence the remarkably good fit between data and model. Figure 3d shows that reconstructed $P_t$ based on the numerically optimal linear exponential filter (red) and the $2/3$ rule (blue) both track the true Bayesian $P_t$ very well.

In the previous section, we saw that exact Bayesian inference for the DBM is a good model of behavioral data. In this section, we saw that linear exponential filtering also seems to capture the data well. To compare which of the two better explains the data, we need a more detailed account of how stimulus history-dependent probabilities translate into reaction times. A growing body of psychological [7] and physiological data [8] support the notion that some form of evidence integration up to a fixed threshold underlies binary perceptual decision making, which both optimizes an accuracy-RT trade-off [9] and seems to be implemented in some form by cortical neurons [8]. The idealized, continuous-time version of this, the drift-diffusion model (DDM), has a well characterized mean stopping time [10], $T_d = \frac{z}{A}\tanh\frac{Az}{c^2}$, where $A$ and $c$ are the mean and standard deviation of unit time fluctuation, and $z$ is the distance between the starting point and decision boundary. The vertical axis for the DDM is in units of log posterior ratio $\log\frac{P(s_0|\mathbf{x}_t)}{P(s_1|\mathbf{x}_t)}$. An unbiased (uniform) prior over $s$ implies a stochastic trajectory that begins at 0 and drifts until it hits one of the two boundaries $\pm z$. When the prior is biased at $b \neq .5$, it has an additive effect in the log posterior ratio space and moves the starting point to $\log\frac{b}{1-b}$. For the relevant range of $b$ (.2 to .8), the shift shift in starting point is approximately linear in $b$ (Figure 3e inset), so that the new distance to the boundary is approximately $z + kb$. Thus, the new mean decision time is $\frac{z+kb}{A}\tanh\frac{Az+Akb}{c^2}$. Typically in DDM models of decision-making, the signal-to-noise ratio is small, i.e. $A \ll c$, such that $\tanh$ is highly linear in the relevant range. We therefore have $T_d(b) \approx \frac{z^2}{c^2} + \frac{2zk}{c^2}b$, implying that the change in mean decision time is linear in the bias $b$, in units of probability.

This linear relationship between RT and $b$ was already born out by the good fit between sequential effects in behavioral data and for the DBM in Figure 1d. To examine this more closely, we run the exact Bayesian DBM algorithm and the linear exponential filter on the actual sequences of stimuli observed by the subjects, and plot median RT against predicted stimulus probabilities. In Figure 3e, we see that for both exact Bayesian (red) and exponential (blue) algorithms, RT's decrease on repetition stimuli when predicted probability for repetition increased; conversely, RT's increase on alternation trials when predicted probability for repetition increase (and therefore predicted probability for alternation decrease). For both Bayesian inference and linear exponential filtering, the relationship between RT and stimulus probability is approximately linear. The linear fit in fact appears better for the exponential algorithm than exact Bayesian inference, which, conditioned on the DDM being an appropriate model for binary decision making, implies that the former may be a better model of sequential adaptation than exact Bayesian inference. Further experimentation is underway to examine this prediction more carefully.

Another implication of the SPRT or DDM formulation of perceptual decision-making is that incorrect prior bias, such as due to sequential effects in a randomized stimulus sequence, induces a net cost in accuracy (even though the RT effects wash out due to the linear dependence on prior bias). The error rate with a bias $x_0$ in starting point is $\frac{1}{1+e^{2za}} - \frac{1-(e^{-ax_0})^2}{e^{2az}-e^{-2az}}$ [10], implying error rate rises monotonically with bias in either direction. This is a quantitative characterization of our claim that extrageneous prior bias, such as due to sequential effects, induces suboptimality in decision-making.

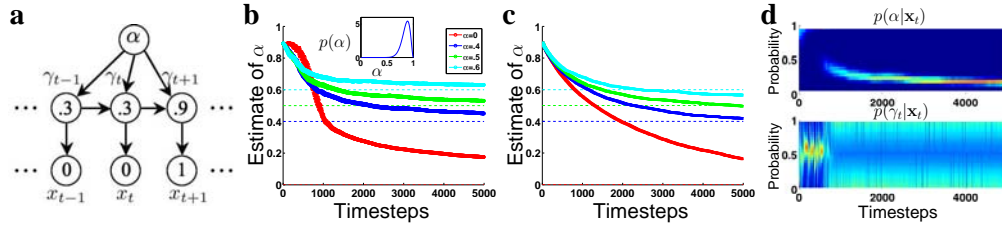

Figure 4: Meta-learning about the rate of change. (a) Graphical model for exact Bayesian learning. Numbers are example values for the variables. (b) Mean of posterior $p(\alpha|\mathbf{x}_t)$ as a function of timesteps, averaged over 30 sessions of simulated data, each set generated from different true values of $\alpha$ (see legend; color-coded dashed lines indicate true $\alpha$). Inset shows prior over $\alpha$, $p(\alpha) = Beta(17,3)$. Time-course of learning is not especially sensitive to the exact form of the prior (not shown). (c) Stochastic gradient descent with a learning rate of .01 produce estimates of $\alpha$ (thick lines, width denotes SEM) that converge to the true values of $\alpha$ (dashed lines). Initial estimate of $\alpha$, before seeing any data, is .9. Learning based on 50 sessions of 5000 trials for each value of $\alpha$. (d) Marginal posterior distributions over $\alpha$ (top panel) and $\gamma_t$ (bottom panel) on a sample run, where probability mass is color-coded: brighter color is more mass.

## 4 Neural implementation and learning

So far, we have seen that exponential discounting of the past not only approximates exact Bayesian inference, but fits human behavioral data. We now note that it has the additional appealing property of being equivalent to standard models of neuronal dynamics. This is because the iterative form of the linear exponential filter in Equation 1 has a similar form to a large class of leaky integration neuronal models, which have been used extensively to model perceptual decision-making on a relatively fast time-scale [8, 11–15], as well as trial-to-trial interactions on a slower time-scale [16–20]. It is also related to the concept of eligibility trace in reinforcement learning [21], which is important for the temporal credit assignment problem of relating outcomes to states or actions that were responsible for them. Here, we provided the *computational rationale* for this exponential discounting the past – it approximates Bayesian inference under DBM-like assumptions.

Viewed as a leaky-integrating neuronal process, the parameters of Equation 1 have the following semantics: $\frac{1}{2}(1-\alpha)$ can be thought of as a constant bias, $\frac{1}{3}\alpha x_{t-1}$ as the feed-forward input, and $\frac{2}{3}\alpha P_{t-1}$ as the leaky recurrent term. Equation 1 suggests that neurons utilizing a standard form of integration dynamics can implement near-optimal Bayesian prediction under the non-stationary assumption, as long as the relative contributions of the different terms are set appropriately. A natural question to ask next is how neurons can *learn* to set the weights appropriately. We first note that $x_t$ is a sample from the distribution $P(x_t|\mathbf{x}_{t-1})$. Since $P(x_t|\mathbf{x}_{t-1})$ has the approximate linear form in Equation 1, with dependence on a single parameter $\alpha$, learning about near-optimal predictions can potentially be achieved based on estimating the value of $\alpha$ via the stochastic samples $x_1, x_2, \ldots$. We implement a stochastic gradient descent algorithm, in which $\hat{\alpha}$ is adjusted incrementally on each trial in the direction of the gradient, which should bring $\hat{\alpha}$ closer to the true $\alpha$.

$$\hat{\alpha}_t = \hat{\alpha}_{t-1} + \epsilon(x_t - \hat{P}_t)\frac{dP_t}{d\alpha}$$

where $\hat{\alpha}_t$ is the estimate of $\alpha$ after observing $x_t$, and $\hat{P}_t$ is the estimate of $P_t$ using the estimate $\hat{\alpha}_{t-1}$ (before seeing $x_t$). Figure 4c shows that learning via the binary samples is indeed possible: for different true values of $\alpha$ (dashed lines) that generated different data sets, stochastic gradient descent produced estimates of $\hat{\alpha}$ that converge to the true values, or close to them (thick lines; widths denote SEM estimated from 50 sessions of learning). A key challenge for future work is to clarify whether and how the gradient, $\frac{dP_t}{d\alpha}$, can be computed by neural machinery (perhaps approximately).

For comparison, we also implement the exact Bayesian learning algorithm, which augments the DBM architecture by representing $\alpha$ as a hidden variable instead of a fixed known parameter:

$$p(\alpha, \gamma_t|\mathbf{x}_t) \propto p(\alpha|\mathbf{x}_{t-1})P(x_t|\gamma_t)p(\gamma_t|\alpha, \mathbf{x}_{t-1}) .$$

Figure 4a illustrates this augmented model graphically. Figure 4b shows the evolution of the mean of the posterior distribution over $\alpha$, or $\langle\alpha|\mathbf{x}_t\rangle$. Based on sets of 30 sessions of 5000 trials, generated

from each of four different true values of $\alpha$, the mean value of $\alpha$ under the posterior distribution tends toward the true $\alpha$ over time. The prior we assume for $\alpha$ is a beta distribution ($Beta(17, 3)$, shown in the inset of Figure 4b).

Compared to exact Bayesian learning, stochastic gradient descent has a similar learning rate. But larger values of $\alpha$ (e.g. $\alpha = .6$) tend to be under-estimated, possibly due to the fact that the analytical approximation for $\beta$ is under-estimated for larger $\alpha$. For data that were generated from a fixed Bernoulli process with rate .5, an equivalently appropriate model is the DBM with $\alpha = 0$ – stochastic gradient descent produced estimates of $\alpha$ (thick red line) that converge to 0 on the order of 50000 trials (details not shown). Figure 4d shows that the posterior inference about $\alpha$ and $\gamma_t$ undergoes distinct phases when true $\alpha = 0$ and there is no correlation between one timestep and the next. There is an initial phase where marginal posterior mass for $\alpha$ tends toward high values of $\alpha$, while marginal posterior mass for $\gamma_t$ fluctuates around .5. Note that this combination is an alternative, equally valid generative model for completely randomized sequence of inputs. However, this joint state is somehow unstable, and $\alpha$ tends toward 0 while $\gamma_t$ becomes broad and fluctuates wildly. This is because as inferred $\alpha$ gets smaller, there is almost no information about $\gamma_t$ from past observations, thus the marginal posterior over $\gamma_t$ tends to be broad (high uncertainty) and fluctuates along with each data point. $\alpha$ can only decrease slowly because so little information about the hidden variables is obtained from each data point. For instance, it is very difficult to infer from what is believed to be an essentially random sequence whether the underlying Bernoulli rate really tends to change once every 1.15 trials or 1.16 trials. This may explain why subjects show no diminished sequential effects over the course of a few hundred trials (Figure 1d). While the stochastic gradient results demonstrate that, in principle, the correct values of $\alpha$ can be learned via the sequence of binary observations $x_1, x_2, \ldots$, further work is required to demonstrate whether and how neurons could implement the stochastic gradient algorithm or an alternative learning algorithm .

## 5 Discussion

Humans and other animals constantly have to adapt their behavioral strategies in response to changing environments: growth or shrinkage in food supplies, development of new threats and opportunities, gross changes in weather patterns, etc. Accurate tracking of such changes allow the animals to adapt their behavior in a timely fashion. Subjects have been observed to readily alter their behavioral strategy in response to recent trends of stimulus statistics, even when such trends are spurious. While such behavior is sub-optimal for certain behavioral experiments, which interleave stimuli randomly or pseudo-randomly, it is appropriate for environments in which changes do take place on a slow timescale. It has been observed, in tasks where statistical contingencies undergo occasional and unsignaled changes, that monkeys weigh past observations linearly but with decaying coefficients (into the past) in choosing between options [6]. We showed that human subjects behave very similarly in 2AFC tasks with randomized design, and that such discounting gives rise to the frequently observed sequential effects found in such tasks [5]. We showed that such exponential discounting approximates *optimal* Bayesian inference under assumptions of statistical non-stationarity, and derived an analytical, approximate relationship between the parameters of the optimal linear exponential filter and the statistical assumptions about the environment. We also showed how such computations can be implemented by leaky integrating neuronal dynamics, and how the optimal tuning of the leaky integration process can be achieved without explicit representation of probabilities.

Our work provides a normative account of *why* exponential discounting is observed in both stationary and non-stationary environments, and *how* it may be implemented neurally. The relevant neural mechanisms seem to be engaged both in tasks when the environmental contingencies are truly changing at unsignaled times, and also in tasks in which the underlying statistics are stationary but chance patterns masquerade as changing statistics (as seen in sequential effects). This work bridges and generalizes previous *descriptive* accounts of behavioral choice under non-stationary task conditions [6], as well as *mechanistic* models of how neuronal dynamics give rise to trial-to-trial interactions such as priming or sequential effects [5, 13, 18–20]. Based the relationship we derived between the rate of behavioral discounting and the subjects' implicit assumptions about the rate of environmental changes, we were able to "reverse-engineer" the subjects' internal assumptions. Subjects appear to assume $\alpha = .77$, or changing about once every four trials. This may have implications for understanding why working memory has the observed capacity of 4-7 items.

In a recent human fMRI study [22], subjects appeared to have different learning rates in two phases of slower and faster changes, but notably the first phase contained *no* changes, while the second phase contained frequent ones. This is a potential confound, as it has been observed that adaptive responses change significantly upon the first switch but then settle into a more stable regime [23]. It is also worth noting that different levels of sequential effects/adaptive response appear to take place at different time-scales [4,23], and different neural areas seem to be engaged in processing different types of temporal patterns [24]. In the context of our model, it may imply that there is sequential adaptation happening at different levels of processing (e.g. sensory, cognitive, motor), and their different time-scales may reflect different characteristic rate of changes at these different levels. A related issue is that brain needs not to have explicit representation of the rate of environmental changes, which are implicitly encoded in the "leakiness" of neuronal integration over time. This is consistent with the observation of sequential effects even when subjects are explicitly told that the stimuli are random [4]. An alternative explanation is that subjects do not have complete faith in the experimenter's instructions [25]. Further work is needed to clarify these issues.

We used both a computationally optimal Bayesian learning algorithm, and a simpler stochastic gradient descent algorithm, to learn the rate of change (1-$\alpha$). Both algorithms were especially slow at learning the case when $\alpha = 0$, which corresponds to truly randomized inputs. This implies that completely random statistics are difficult to internalize, when the observer is searching over a much larger hypothesis space that contains many possible models of statistical regularity, which can change over time. This is consistent with previous work [26] showing that discerning "randomness" from binary observations may require surprisingly many samples, when statistical regularities are presumed to change over time. Although this earlier work used a different model for what kind of statistical regularities are allowed, and how they change over time (temporally causal and Markovian in ours, an acausal correlation function in theirs), as well as the nature of the inference task (on-line in our setting, and off-line in theirs), the underlying principles and conclusions are similar: it is very difficult to discriminate a truly randomized sequence, which by chance would contain runs of repetitions and alternations, from one that has *changing biases* for repetitions and alternations over time.

## References

[1]  Skinner, B F (1948). *J. Exp. Psychol.* **38**: 168-72.

[2]  Ecott, C L & Critchfield, T S (2004). *J. App. Beh. Analysis* **37**: 249-65.

[3]  Laming, D R J (1968). *Information Theory of of Choice-Reaction Times*, Academic Press, London.

[4]  Soetens, E, Boer, L C, & Hueting, J E (1985). *JEP: HPP* **11**: 598-616.

[5]  Cho, R, et al (2002). *Cognitive, Affective, & Behavioral Neurosci.* **2**: 283-99.

[6]  Sugrue, L P, Corrado, G S, & Newsome, W T (2004). *Science* **304**: 1782-7.

[7]  Smith, P L & Ratcliff, R. *Trends Neurosci.* **27**: 161-8.

[8]  Gold, J I & Shadlen, M N (2002). *Neuron* **36**: 299-308.

[9]  Wald, A & Wolfowitz, J (1948). *Ann. Math. Statisti.* **19**: 326-39.

[10]  Bogacz, et al (2006). *Psychological Review* **113**: 700-65.

[11]  Cook, E P & Maunsell, J H R (2002). *Nat. Neurosci.* **5**: 985-94.

[12]  Grice, G R (1972). *Perception & Psychophysics* **12**: 103-7.

[13]  McClelland, J L. *Attention & Performance XIV*: 655-88. MIT Press.

[14]  Smith, P L (1995). *Psychol. Rev.* **10**: 567-93.

[15]  Yu, A J (2007). *Adv. in Neur. Info. Proc. Systems* **19**: 1545-52.

[16]  Dayan, P & Yu, A J (2003). *IETE J. Research* **49**: 171-81.

[17]  Kim, C & Myung, I J (1995). *17th Ann. Meeting. of Cog. Sci. Soc.*: 472-7.

[18]  Mozer, M C, Colagrosso, M D, & Huber, D E (2002). *Adv. in Neur. Info. Proc. Systems* **14**: 51-57.

[19]  Mozer, M C, Kinoshita, S, & Shettel, M (2007). *Integrated Models of Cog. Sys.*: 180-93.

[20]  Simen, P, Cohen, J D, & Holmes, P (2006). *Neur. Netw.* **19**: 1013-26.

[21]  Sutton, R S & Barto, A G (1998). *Reinforcement Learning: An Introduction*, MIT Press.

[22]  Behrens, T E J, Woolrich, M W, Walton, M E, & Rushworth, M F S (2007). *Nat. Neurosci.* **10**: 1214-21.

[23]  Kording, K P, Tenenbaum, J B, & Shadmehr, R (2007). *Nat. Neurosci.* **10**: 779-86.

[24]  Huettel, S A, Mack, P B, & McCarthy, G (2002). *Nat. Neurosci.* **5**: 485-90.

[25]  Hertwig, R & Ortmann, A (2001). *Behavioral & Brain Sciences* **24**: 383-403.

[26]  Bialek, W (2005). Preprint q-bio.NC/0508044, Princeton University.

